# Modelling motion primitives and their timing in biologically executed movements

**Ben H Williams**
School of Informatics
University of Edinburgh
5 Forrest Hill, EH1 2QL, UK
ben.williams@ed.ac.uk

**Marc Toussaint**
TU Berlin
Franklinstr. 28/29, FR 6-9
10587 Berlin, Germany
mtoussai@cs.tu-berlin.de

**Amos J Storkey**
School of Informatics
University of Edinburgh
5 Forrest Hill, EH1 2QL, UK
a.storkey@ed.ac.uk

## Abstract

Biological movement is built up of sub-blocks or *motion primitives*. Such primitives provide a compact representation of movement which is also desirable in robotic control applications. We analyse handwriting data to gain a better understanding of primitives and their timings in biological movements. Inference of the shape and the timing of primitives can be done using a factorial HMM based model, allowing the handwriting to be represented in primitive timing space. This representation provides a distribution of spikes corresponding to the primitive activations, which can also be modelled using HMM architectures. We show how the coupling of the low level primitive model, and the higher level timing model during inference can produce good reconstructions of handwriting, with shared primitives for all characters modelled. This coupled model also captures the variance profile of the dataset which is accounted for by spike timing jitter. The timing code provides a compact representation of the movement while generating a movement without an explicit timing model produces a scribbling style of output.

## 1 Introduction

Movement planning and control is a very difficult problem in real-world applications. Current robots have very good sensors and actuators, allowing accurate movement execution, however the ability to organise complex sequences of movement is still far superior in biological organisms, despite being encumbered with noisy sensory feedback, and requiring control of many non-linear and variable muscles. The underlying question is that of the representation used to generate biological movement. There is much evidence to suggest that biological movement generation is based upon *motor primitives*, with discrete muscle synergies found in frog spines, (Bizzi et al., 1995; d'Avella & Bizzi, 2005; d'Avella et al., 2003; Bizzi et al., 2002), evidence of primitives being locally fixed (Kargo & Giszter, 2000), and modularity in human motor learning and adaption (Wolpert et al., 2001; Wolpert & Kawato, 1998). Compact forms of representation for any biologically produced data should therefore also be based upon primitive sub-blocks.

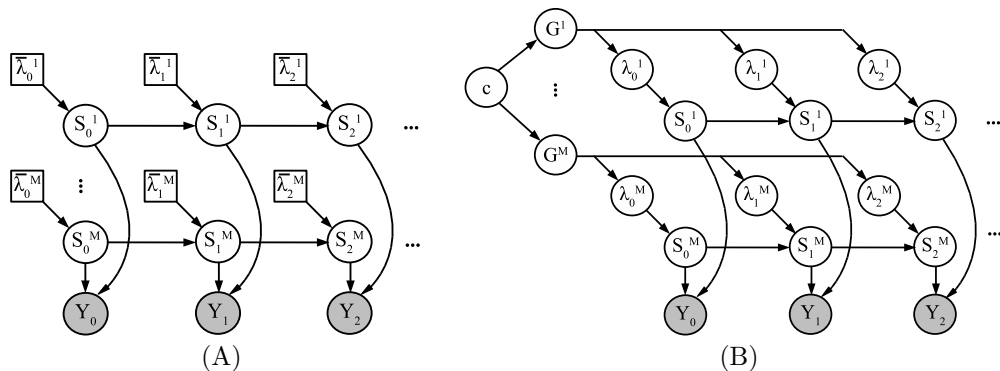

Figure 1: (A) A factorial HMM of a handwriting trajectory $Y_t$. The parameters $\bar{\lambda}_t^m$ indicate the probability of triggering a primitive in the $m^{th}$ factor at time $t$ and are learnt for one specific character. (B) A hierarchical generative model of handwriting where the random variable $c$ indicates the currently written character and defines a distribution over random variables $\lambda_t^m$ via a Markov model over $G^m$.

There are several approaches to use this idea of motion primitives for more efficient robotic movement control. (Ijspeert et al., 2003; Schaal et al., 2004) use non-linear attractor dynamics as a motion primitive and train them to generate motion that solves a specific task. (Amit & Matarić, 2002) use a single attractor system and generate non-linear motion by modulating the attractor point. These approaches define a primitive as a segment of movement rather than understanding movement as a *superposition* of concurrent primitives. The goal of analysing and better understanding biological data is to extract a *generative* model of complex movement based on concurrent primitives which may serve as an efficient representation for robotic movement control. This is in contrast to previous studies of handwriting which usually focus on the problem of character classification rather than generation (Singer & Tishby, 1994; Hinton & Nair, 2005).

We investigate handwriting data and analyse whether it can be modelled as a superposition of sparsely activated motion primitives. The approach we take can intuitively be compared to a Piano Model (also called Piano roll model (Cemgil et al., 2006)). Just as piano music can (approximately) be modelled as a superposition of the sounds emitted by each key we follow the idea that biological movement is a superposition of pre-learnt motion primitives. This implies that the whole movement can be compactly represented by the timing of each primitive in analogy to a score of music. We formulate a probabilistic generative model that reflects these assumptions. On the lower level a factorial Hidden Markov Model (fHMM, Ghahramani & Jordan, 1997) is used to model the output as a combination of signals emitted from independent primitives (each primitives corresponds to a factor in the fHMM). On the higher level we formulate a model for the primitive timing dependent upon character class. The same motion primitives are shared across characters, only their timings differ. We train this model on handwriting data using an EM-algorithm and thereby infer the primitives and the primitive timings inherent in this data. We find that the inferred timing posterior for a specific character is indeed a compact representation for the specific character which allows for a good reproduction of this character using the learnt primitives. Further, using the timing model learnt on the higher level we can generate new movement – new samples of characters (in the same writing style as the data), and also scribblings that exhibit local similarity to written characters when the higher level timing control is omitted.

Section 2 will introduce the probabilistic generative model we propose. Section 3 briefly describes the learning procedures which are variants of the EM-algorithm adapted to our model. Finally in section 4 we present results on handwriting data recorded with a digitisation tablet, show the primitives and timing code we extract, and demonstrate how the learnt model can be used to generate new samples of characters.

## 2 Model

Our analysis of primitives and primitive timings in handwriting is based on formulating a corresponding probabilistic generative model. This model can be described on two levels. On the lower level (Figure 1(A)) we consider a factorial Hidden Markov Model (fHMM) where each factor produces the signal of a single primitive and the linear combination of factors generates the observed movement $Y_t$. This level is introduced in the next section and was already considered in (Williams et al., 2006; Williams et al., 2007). It allows the learning and identification of primitives in the data but does not include a model of their timing. In this paper we introduce the full generative model (Figure 1(B)) which includes a generative model for the primitive timing conditioned on the current character.

### 2.1 Modelling primitives in data

Let $M$ be the number of primitives we allow for. We describe a primitive as a strongly constrained Markov process which remains in a zero state most of the time but with some probability $\bar{\lambda} \in [0, 1]$ enters the 1 state and then rigorously runs through all states $2, .., K$ before it enters the zero state again. While running though its states, this process emits a fixed temporal signal. More rigorously, we have a fHMM composed of $M$ factors. The state of the $m^{th}$ factor at time $t$ is $S_t^m \in \{0, .., K_m\}$, and the transition probabilities are

$$P(S_t^m = b \mid S_{t-1}^m = a, \bar{\lambda}_t^m) = \begin{cases} \bar{\lambda}_t^m & \text{for } a = 0 \text{ and } b = 1 \\ 1 - \bar{\lambda}_t^m & \text{for } a = 0 \text{ and } b = 0 \\ 1 & \text{for } a \neq 0 \text{ and } b = (a + 1) \bmod K_m \\ 0 & \text{otherwise} \end{cases} . \tag{1}$$

This process is parameterised by the onset probability $\bar{\lambda}_t^m$ of the $m^{th}$ primitive at time $t$. The $M$ factors emit signals which are combined to produce the observed motion trajectory $Y_t$ according to

$$P(Y_t \mid S_t^{1:M}) = \mathcal{N}(Y_t, \sum_{m=1}^{M} W_{S_t^m}^m, C) , \tag{2}$$

where $\mathcal{N}(x, a, A)$ is the Gaussian density function over $x$ with mean $a$ and covariance matrix $A$. This emission is parameterised by $W_s^m$ which is constrained to $W_0^m = 0$ (the zero state does not contribute to the observed signal), and $C$ is a stationary output covariance.

The vector $W_{1:K_m}^m = (W_1^m, .., W_{K_m}^m)$ is what we call a primitive and – to stay in the analogy – can be compared to the sound of a piano key. The parameters $\bar{\lambda}_t^m \in [0, 1]$ could be compared to the score of the music. We will describe below how we learn the primitives $W_s^m$ and also adapt the primitive lengths $K_m$ using an EM-algorithm.

### 2.2 A timing model

Considering the $\bar{\lambda}$'s to be fixed parameters is not a suitable model of biological movement. The usage and timing of primitives depends on the character that is written and the timing varies from character to character. Also, the $\bar{\lambda}$'s actually provide a rather high-dimensional representation for the movement. Our model takes a different approach to parameterise the primitive activations. For instance, if a primitive is activated twice in the course of the movement we assume that there have been two signals ("spikes") emitted from a higher level process which encode the activation times. More formally, let $c$ be a discrete random variable indicating the character to be written, see Figure 1(B). We assume that for each primitive we have another Markovian process which generates a length-$L$ sequence of states $G_l^m \in \{1, .., R, 0\}$,

$$P(G_{1:L}^m \mid c) = P(G_1^m \mid c) \prod_{l=2}^{L} P(G_l^m \mid G_{l-1}^m, c) . \tag{3}$$

The states $G_l^m$ encode which primitives are activated and how they are timed, as seen in Figure 2(b). We now define $\lambda_t^m$ to be a binary random variable that indicate the activation

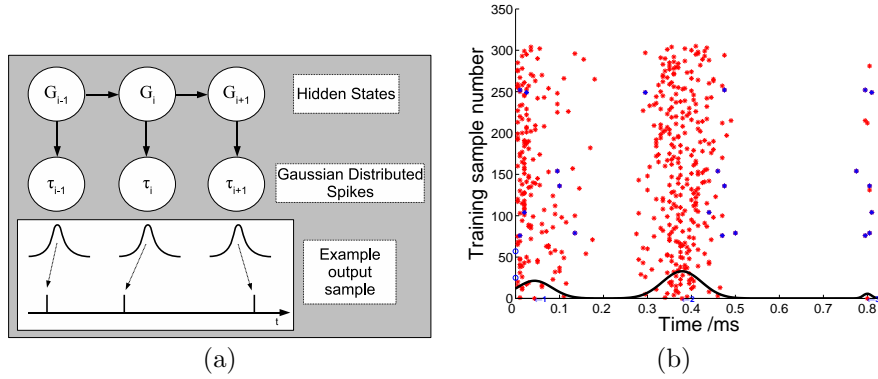

Figure 2: (a) Illustration of equation (4): The Markov process on the states $G_l^m$ emits Gaussian components to the onset probabilities $P(\lambda_t^m = 1)$. (b) Scatter plot of the MAP onsets of a single primitive for different samples of the same character 'p'. Gaussian components can be fit to each cluster.

of a primitive at time $t$, which we call a "spike". For a zero-state $G_l^m = 0$ no spike is emitted and thus the probability of $\lambda^m = 1$ is not increased. A non-zero state $G_l^m = r$ adds a Gaussian component to the probabilities of $\lambda_t^m = 1$ centred around a typical spike time $\mu_r^m$ and with variance $\sigma_r^m$,

$$P(\lambda_t^m = 1 \mid G_{1:K_m}^m, c) = \sum_{l=1}^{L} \delta_{G_l^m > 0} \int_{t-0.5}^{t+0.5} \mathcal{N}(t, \mu_{G_l^m}^m, \sigma_{G_l^m}^m) \, dt \ . \tag{4}$$

Here, $\delta_{G_l^m > 0}$ is zero for $G_l^m = 0$ and 1 otherwise, and the integral essentially discretises the Gaussian density. Additionally, we restrict the Markovian process such that each Gaussian component can emit at most one spike, i.e., we constrain $P(G_l^m \mid G_{l-1}^m, c)$ to be a lower triangular matrix. Given the $\lambda$'s, the state transitions in the fHMM factors are as in equation (1), replacing $\bar{\lambda}$ by $\lambda$.

To summarise, the spike probabilities of $\lambda_t^m = 1$ are a sum of at most $L$ Gaussian components centred around the means $\mu_l^m$ and with variances $\sigma_l^m$. Whether or not such a Gaussian component is present is itself randomised and depends on the states $G_l^m$. We can observe at most $L$ spikes in one primitive, the spike times between different primitives are dependent, but we have a Markovian dependency between the presence and timing of spikes within a primitive. The whole process is parameterised by the initial state distribution $P(G_1^m \mid c)$, the transition probabilities $P(G_l^m \mid G_{l-1}^m, c)$, the spike means $\mu_r^m$ and the variances $\sigma_r^m$. All these parameters will be learnt using an EM-algorithm.

This timing model is motivated from results with the fHMM-only model: When training the fHMM on data of a single character and then computing the MAP spike times using a Viterbi alignment for each data sample we find that the MAP spike times are roughly Gaussian distributed around a number of means (see Figure 2(b)). This is why we used a sum of Gaussian components to define the onset probabilities $P(\lambda = 1)$. However, the data is more complicated than provided for by a simple Mixture of Gaussians. Not every sample includes an activation for each cluster (which is a source of variation in the handwriting) and there cannot be more than one spike in each cluster. Therefore we introduced the constrained Markov process on the states $G_l^m$ which may skip the emission of some spikes.

## 3   Inference and learning

In the experiments we will compare both the fHMM without the timing model (Figure 1(A)) and the full model including the timing model (Figure 1(B)).

In the fHMM-only model, inference in the fHMM is done using variational inference as described in (Ghahramani & Jordan, 1997). Using a standard EM-algorithm we can train the parameters $W$, $C$ and $\bar{\lambda}$. To prevent overfitting we assume the spike probabilities

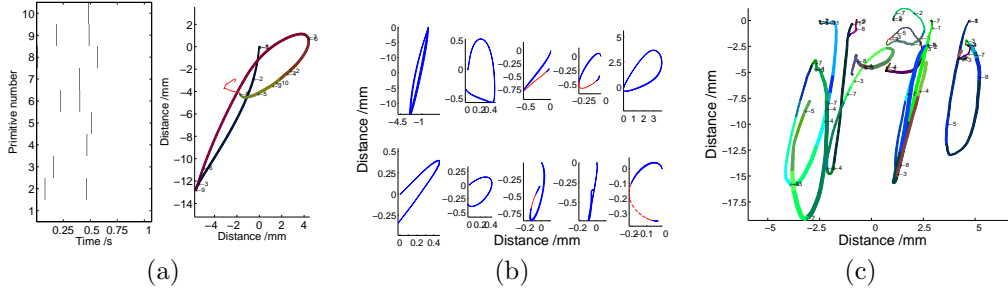

Figure 3: (a) Reconstruction of a character from a training dataset, using a subset of the primitives. The thickness of the reconstruction represents the pressure of the pen tip, and the different colours represent the activity of the different primitives, the onsets of which are labelled with an arrow. The posterior probability of primitive onset is shown on the left, highlighting why a spike timing representation is appropriate. (b) Plots of the 10 extracted primitives, as drawn on paper. (c) Generative samples using a flat primitive onset prior, showing scribbling behaviour of uncoupled model.

are stationary ($\lambda_t^m$ constant over $t$) and learn only a single mean parameter $\bar{\lambda}^m$ for each primitive.

In the full model, inference is an iterative process of inference in the timing model and inference in the fHMM. Note that variational inference in the fHMM is itself an iterative process which recomputes the posteriors over $S_t^m$ after adapting the variational parameters. We couple this iteration to inference in the timing model in both directions: In each iteration, the posterior over $S_t^m$ defines observation likelihoods for inference in the Markov models $G_l^m$. Inversely, the resulting posterior over $G_l^m$ defines a new prior over $\lambda$'s (a message from $G_l^m$ to $\lambda_t^m$) which enter the fHMM inference in the next iteration. Standard M-steps are then used to train all parameters of the fHMM and the timing model. In addition, we use heuristics to adapt the length $K_m$ of each primitive: we increase or decrease $K_m$ depending on whether the learnt primitive is significantly different to zero in the last time steps. The number of parameters used in the model therefore varies during learning, as the size of $W$ depends upon $K_m$, and the size of $G$ depends upon the number of inferred spikes.

In the experiments we will also investigate the reconstruction of data. By this we mean that we take a trained model, use inference to compute the MAP spikes $\lambda$ for a specific data sample, then we use these $\lambda$'s and the definition of our generative model (including the learnt primitives $W$) to generate a trajectory which can be compared to the original data sample. Such a reconstruction can be computed using both the fHMM-only model and the full model.

## 4    Results

### 4.1    Primitive and timing analysis using the fHMM-only

We first consider a data set of 300 handwritten 'p's recorded using an INTUOS 3 WACOM digitisation tablet http://www.wacom.com/productinfo/9x12.cfm, providing trajectory data at 200Hz. The trajectory $Y_t$ we model is the normalised first differential of the data, so that the data mean was close to zero, providing the requirements for the zero state assumption in the model constraints. Three dimensional data was used, x-position, y-position, and pressure. The data collected were separated into samples, or characters, allowing each sample to be separately normalised.

Our choice of parameter was $M = 10$ primitives and we initialised all $K_m = 20$ and constrained them to be smaller than 100 throughout learning.

We trained the fHMM-only model on this dataset. Figure 3(a) shows the reconstruction of a specific sample of this data set and the corresponding posterior over $\lambda$'s. This clean posterior is the motivation for introducing a model of the spike timings as a compact representation

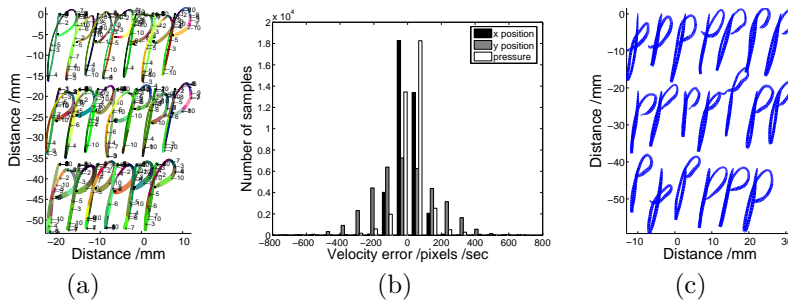

$$(a) \qquad\qquad\qquad (b) \qquad\qquad\qquad (c)$$

Figure 4: (a) Reconstructions of 'p's using the full model. (b) Histogram of the reconstruction error, which is 3-dimensional pen movement velocity space. These errors were produced using over 300 samples of a single character. (c) Generative samples using the full generative model (Figure 1(B)).

of the data. Equally the reconstruction (using the Viterbi aligned MAP spikes) shows the sufficiency of the spike code to generate the character. Figure 3(b) shows the primitives $W^m$ (translated back into pen-space) that were learnt and implicitly used for the reconstruction of the 'p'. These primitives can be seen to represent typical parts of the 'p' character; the arrows in the reconstruction indicate when they are activated.

The fHMM-only model can be used to reconstruct a specific data sample using the MAP $\lambda$'s of that sample, but it can not 'autonomously' produce characters since it lacks a model of the timing. To show the importance of this spike timing information, we can demonstrate the effects of removing it. When using the fHMM-only model as a generative model with the learnt stationary spike probabilities $\bar{\lambda}^m$ the result is a form of *primitive babbling*, as can be seen in Figure 3(c). Since these scribblings are generated by random expression of the learnt primitives they locally resemble parts of the 'p' character.

The primitives generalise to other characters if the training dataset contained sufficient variation. Further investigation has shown that 20 primitives learnt from 12 character types are sufficiently generalised to represent all remaining novel character types without further learning, by using a single E-step to fit the pre-learnt parameters to a novel dataset.

## 4.2   Generating new characters using the full generative model

Next we trained the full model on the same 'p'-dataset. Figure 4(a) shows the reconstructions of some samples of the data set. To the right we see the reconstruction errors in velocity space showing at many time points a perfect reconstruction was attained. Since the full model includes a timing model it can also be run autonomously as a generative model for new character samples. Figure 4(c) displays such new samples of the character 'p' generated by the learnt model.

As a more challenging problem we collected a data set of over 450 character samples of the letters $a$, $b$ and $c$. The full model includes the written character class as a random variable and can thus be trained on multi-character data sets. Note that we restrict the total number of primitives to $M = 10$ which will require a sharing of primitives across characters. Figure 5(a) shows samples of the training data set while Figure 5(b) shows reconstructions of the same samples using the MAP $\lambda$'s in the full model. Generally, the reconstructions using the full model are better than using the fHMM-only model. This can be understood investigating the distribution of the MAP $\lambda$'s across different samples under the fHMM-only and the full model, see Figure 6. Coupling the timing and the primitive model during learning has the effect of trying to learn primitives from data that are usually in the same place. Thus, using the full model the inferred spikes are more compactly clustered at the Gaussian components due to the prior imposed from the timing model (the thick black lines correspond to Equation (4)).

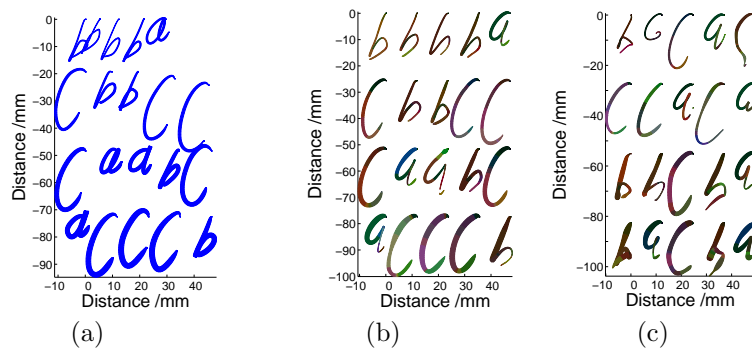

(a)  (b)  (c)

Figure 5: (a) Training dataset, showing 3 character types, and variation. (b) Reconstruction of dataset using 10 primitives learnt from the dataset in (a). (c) Generative samples using the full generative model (Figure 1(B)).

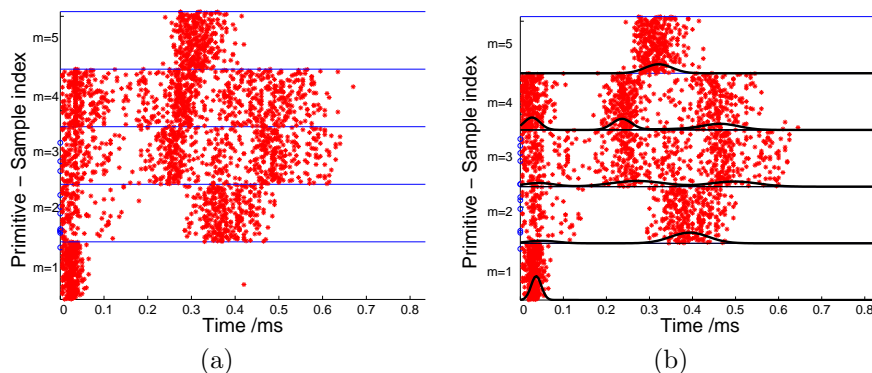

(a)  (b)

Figure 6: (a) Scatter plot of primitive onset spikes for a single character type across all samples and primitives, showing the clustering of certain primitives in particular parts of a character. The horizontal bars separate the results for different primitives. (b) Scatter plot of spikes from same dataset, with a coupled model, showing suppression of outlying spikes and tightening of clusters. The thick black lines displays the prior over $\lambda$'s imposed from the timing model via Equation (4).

Finally, we run the full model autonomously to generate new character samples, see Figure 5(c). Here the character class, $c$ is first sampled uniform randomly and then all learnt parameters are used to eventually sample a trajectory $Y_t$. The generative samples show interesting variation while still being readably a character.

## 5    Conclusions

In this paper we have shown that it is possible to represent handwriting using a primitive based model. The model consists of a superposition of several arbitrary fixed functions. These functions are time-extended, of variable length (during learning), and are superimposed with learnt offsets. The timing of activations is crucial to the accurate reproduction of the character. With a small amount of timing variation, a distorted version of the original character is reproduced, whilst large (and coordinated) differences in the timing pattern produce different character types.

The spike code provides a compact representation of movement, unlike that which has previously been explored in the domain of robotic control. We have proposed to use Markov processes conditioned on the character as a model for these spike emissions. Besides contributing to a better understanding of biological movement, we hope that such models will inspire applications also in robotic control, e.g., for movement optimisation based on spike codings.

An assumption made in this work is that the primitives are learnt velocity profiles. We have not included any feedback control systems in the primitive production, however the presence of low-level feedback, such as in a spring system (Hinton & Nair, 2005) or dynamic motor primitives (Ijspeert et al., 2003; Schaal et al., 2004), would be interesting to incorporate into the model, and could perhaps be done by changing the outputs of the fHMM to parameterise the spring systems rather than be Gaussian distributions of velocities.

We make no assumptions about how the primitives are learnt in biology. It would be interesting to study the evolution of the primitives during human learning of a new character set. As humans become more confident at writing a character, the reproduction becomes faster, and more repeatable. This could be related to a more accurate and efficient use of primitives already available. However, it might also be the case that new primitives are learnt, or old ones adapted. More research needs to be done to examine these various possibilities of how humans learn new motor skills.

### Acknowledgements

Marc Toussaint was supported by the German Research Foundation (DFG), Emmy Noether fellowship TO 409/1-3.

# References

Amit, R., & Matarić, M. (2002). Parametric primitives for motor representation and control. *Proc. of the Int. Conf. on Robotics and Automation (ICRA)* (pp. 863–868).

Bizzi, E., d'Avella, A., Saltiel, P., & Trensch, M. (2002). Modular organization of spinal motor systems. *The Neuroscientist*, *8*, 437–442.

Bizzi, E., Giszter, S., Loeb, E., Mussa-Ivaldi, F., & Saltiel, P. (1995). Modular organization of motor behavior in the frog's spinal cord. *Trends in Neurosciences*, *18*, 442–446.

Cemgil, A., Kappen, B., & Barber, D. (2006). A generative model for music transcription. *IEEE Transactions on Speech and Audio Processing*, *14*, 679–694.

d'Avella, A., & Bizzi, E. (2005). Shared and specific muscle synergies in natural motor behaviors. *PNAS*, *102*, 3076–3081.

d'Avella, A., Saltiel, P., & Bizzi, E. (2003). Combinations of muscle synergies in the construction of a natural motor behavior. *Nature Neuroscience*, *6*, 300–308.

Ghahramani, Z., & Jordan, M. (1997). Factorial hidden Markov models. *Machine Learning*, *29*, 245–275.

Hinton, G. E., & Nair, V. (2005). Inferring motor programs from images of handwritten digits. *Advances in Neural Information Processing Systems 18 (NIPS 2005)* (pp. 515–522).

Ijspeert, A. J., Nakanishi, J., & Schaal, S. (2003). Learning attractor landscapes for learning motor primitives. *Advances in Neural Information Processing Systems 15 (NIPS 2003)* (pp. 1523–1530). MIT Press, Cambridge.

Kargo, W., & Giszter, S. (2000). Rapid corrections of aimed movements by combination of force-field primitives. *J. Neurosci.*, *20*, 409–426.

Schaal, S., Peters, J., Nakanishi, J., & Ijspeert, A. (2004). Learning movement primitives. *ISRR2003*.

Singer, Y., & Tishby, N. (1994). Dynamical encoding of cursive handwriting. *Biol.Cybern.*, *71*, 227–237.

Williams, B., M.Toussaint, & Storkey, A. (2006). Extracting motion primitives from natural handwriting data. *Int. Conf. on Artificial Neural Networks (ICANN)* (pp. 634–643).

Williams, B., M.Toussaint, & Storkey, A. (2007). A primitive based generative model to infer timing information in unpartitioned handwriting data. *Int. Jnt. Conf. on Artificial Intelligence (IJCAI)* (pp. 1119–1124).

Wolpert, D. M., Ghahramani, Z., & Flanagan, J. R. (2001). Perspectives and problems in motor learning. *TRENDS in Cog. Sci.*, *5*, 487–494.

Wolpert, D. M., & Kawato, M. (1998). Multiple paired forward and inverse models for motor control. *Neural Networks*, *11*, 1317–1329.

